# $\theta$-MRF: Capturing Spatial and Semantic Structure in the Parameters for Scene Understanding

**Congcong Li,   Ashutosh Saxena,   Tsuhan Chen**
Cornell University, Ithaca, NY 14853, United States
cl758@cornell.edu, asaxena@cs.cornell.edu, tsuhan@ece.cornell.edu

## Abstract

For most scene understanding tasks (such as object detection or depth estima-
tion), the classifiers need to consider contextual information in addition to the
local features. We can capture such contextual information by taking as input
the features/attributes from *all* the regions in the image. However, this contextual
dependence also varies with the spatial location of the region of interest, and we
therefore need a different set of parameters for each spatial location. This results
in a very large number of parameters. In this work, we model the independence
properties between the parameters for each location and for each task, by defin-
ing a Markov Random Field (MRF) over the parameters. In particular, two sets
of parameters are encouraged to have similar values if they are spatially close or
semantically close. Our method is, in principle, complementary to other ways
of capturing context such as the ones that use a graphical model over the labels
instead. In extensive evaluation over two different settings, of multi-class object
detection and of multiple scene understanding tasks (scene categorization, depth
estimation, geometric labeling), our method beats the state-of-the-art methods in
all the four tasks.

## 1   Introduction

Most scene understanding tasks (e.g., object detection, depth estimation, etc.) require that we exploit
contextual information in addition to the local features for predicting the labels. For example, a
region is more likely to be labeled as a car if the region below is labeled as road. I.e., we have
to consider information in a larger area around the region of interest. Furthermore, the location of
the region in the image could also have a large effect on its label, and on how it depends on the
neighboring regions. For example, one would look for sky or clouds when looking for an airplane;
however if one sees grass or a runway, then there may still be an airplane (e.g., when the airplane is
on the ground)—here the contextual dependence of the airplane classifier changes based on object's
location in the image.

We can capture such contextual information by using features from *all* the regions in the image, and
then also train a specific classifier of each spatial location for each object category. However, the
dimensionality of the feature space would become quite large,[1] and training a classifier with limited
training data would not be effective. In such a case, one could reduce the amount of context captured
to prevent overfitting. For example, some recent works [22, 33, 37] use context by encoding input
features, but are limited by the amount of context area they can handle.

In our work, we do not want to eliminate the amount of context captured. We therefore keep the large
number of parameters, and model the interaction between the parameters of the classifiers at different
locations and different tasks. For example, the parameters of two neighboring locations are similar.
The key contribution of our work is to note that two parameters may not ascribe a directionality to
the interaction between them. These interactions are sparse, and we represent these interactions as
an undirected graph where the nodes represent the parameters for each location (for each task) and

gories which may occur in any spatial location in the image. Even if we group the regions into 64 ($8 \times 8$) spatial
locations, the total number of parameters will be $107 * 64 * K$ (for $K$ features each). This is rather large, e.g.,
in our multi-class object detection task this number would be about $47.6$ million (see Section 4).

the edges represent the interaction between the parameters. We call this representation a $\theta$-*MRF*, i.e., a *Markov Random Field over the parameters*. This idea is, in principle, complementary to previous works that capture context by capturing the correlation between the labels. Note that our goal is not to directly compare against such models. Instead, we want to answer the question: *How far can we go with just modeling the interactions between the parameters?*

The edges in our $\theta$-MRF not only connect spatial neighbors but also semantic neighbors. In particular, if two tasks are highly correlated, their parameters given to the same image context should be similar. For example, oven is often next to the dishwasher (in a kitchen scene), therefore they should share similar context, indicating that they can share their parameters. These semantic interactions between the parameters from different tasks also follow the undirected graph. Just like object labels are often modeled as conditionally independent of other non-contextual objects given the important context, the corresponding parameters can also be modeled similarly.

There has been a large body of work that capture contextual information in many different ways which are often complementary to ours. These methods range from capturing the correlation between labels using a graphical model to introduce different types of priors on the labels (based on location, prior knowledge, etc.). For example, a graphical model (directed or undirected) is often used to model the dependency between different labels [29, 40, 19, 17]. Informative priors on the labels are also commonly used to improve performance (e.g., [47]). Some previous works enforce priors on the parameters as a directed graph [46, 32], but our model offers a different and perhaps a more relevant perspective than a directed model, in terms of the independence properties modeled.

We extensively evaluate our method on two different settings. First, we consider the task of labeling 107 object categories in the SUN09 dataset, and show that our method gets better performance than the state-of-the-art methods even when with simple regression as the learning model. Second, we consider the multiple tasks of scene categorization, depth estimation and geometry labeling, and again show that our method gets comparable or better performance than the state-of-the-art methods when we use our method with simple regression. Furthermore, we show that our performance is much higher as compared to just using other methods of putting priors on the parameters.

## 2   Related Work

There is a large body of work that leverages contextual information. We possibly cannot do justice to literature, but we mention a few here. Various sources of context have been explored, ranging from the global scene layout, interactions between regions to local features. To incorporate scene-level information, Torralba et al. [47] use the statistics of low-level features across the entire scene to prime object detection. Hoiem et al. [24] and Saxena et al. [45] use 3D scene information to provide priors on potential object locations. Li et al. [32] propose a hierarchical model to make use of contextual information between tasks on different levels. There are also generic approaches [22, 31] that leverage related tasks to boost the overall performance, without requiring considerate insight into specific tasks.

Many works also model context to capture the local interactions between neighboring regions [23, 35, 28], objects [48, 14], or both [16, 10, 2]. Object co-occurence statistics have also been captured in several ways, e.g., using a CRF [40, 19, 17]. Desai et al. [9] combine individual classifiers by considering spatial interactions between the object detections, and solve a unified multi-class object detection problem through a structured discriminative approach. Other ways to share information across categories include sharing representations [12, 30], sharing training examples between categories [36, 15], sharing parameters [26, 27], and so on. Our work lies in the category of sharing parameters, aiming at capturing the dependencies in the parameters for relevant vision applications.

There are several regularization methods when the number of parameters is quite large, e.g., based on L2 norms [6] and Lasso shrinkage methods [42]. Liang et al. [34] present an asymptotic analysis of smooth regularizers. Recent works [26, 1, 18, 25] place interesting priors on parameters. Jalali et al. [26] do multi-task learning by expressing the parameters as a sum of two parts: shared and specific to the task, which combines the $l_\infty$ penalty and $l_1$ penalty to get block-sparse and element-wise sparse components in the parameters. Negahban and Wainright [38] provide analysis of when $l_{1,\infty}$ norm could be useful. Kim and Xing [27] use a tree to construct the hierarchy of multi-task outputs, and then use the tree-guided group lasso to regularize the multi-task regression. In contemporary work [43], Salakhutdinov et al. learn a hierarchy to share the hierarchical parameters for the object appearance models. Our work is motivated by this direction of work, and our focus is to capture spatial and semantic sharing in parameters using undirected graphical models that have appropriate independence properties.

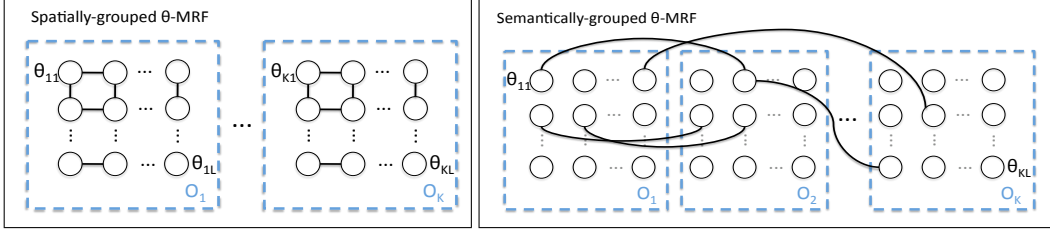

Figure 1: The proposed $\theta$-MRF graph with spatial and semantic interaction structure.

Bayesian priors over parameters are also quite commonly used. For example, [3] uses Dirichlet priors for parameters of a multinomial and normal distribution respectively. In fact, there is a huge body of work on using non-informative priors distributions over parameters [4]—this is particularly useful when the amount of data is not enough to train the parameters. If all the distributions involved (including the prior distribution) are Gaussian, the parameters follow certain useful statistical hyper Markov properties [41, 21, 8]. In applications, [46] considers capturing relationships between the object categories using a Dirichlet prior on the parameters. [20] considers putting posterior sparsity on the parameters instead of parameter sparsity. [11] present a method to learn hyperparameters for CRF-type models. Most of these methods express the prior as another distribution with hyper-parameters—one can view this as a directed graphical model over the parameters. On the other hand, we express relationships between two parameters of the distribution, which does not necessarily involve hyper parameters. This also allows us to capture interesting independence properties.

## 3   Our Approach: $\theta$-MRF

In order to give better intuition, we use the multi-class object detection task as an illustrative example. (Later we will describe and apply it to other scene understanding problems.) Let us consider the $K$-class object detection. We uniformly divide an image into $L$ grids. We then have a binary classifier, whose output is $y_{k,\ell}^{(n)} \in \{0,1\}$ that indicates the presence of the $k^{th}$ object at the $\ell^{th}$ grid in the $n^{th}$ image. Let $x^{(n)}$ be the features (or attributes) extracted from $n^{th}$ image, and let the parameters of the classifier be $\theta_{k,\ell}$. Let $\Theta_k = (\theta_{k,1}, \cdots, \theta_{k,L})$ and let $\boldsymbol{\Theta}$ be the set $\{\Theta_k\}$, $k = 1, \ldots, K$.

Let $P(y_{k,\ell}|x^{(n)}, \theta_{k,\ell})$ be the probability of the output given the input features and the parameters. In order to find the classifier parameters, one typically solves an optimization problem, such as:

$$\underset{\boldsymbol{\Theta}}{\text{minimize}} \qquad \sum_{n}\sum_{k,l} -\log P(y_{k,\ell}|x^{(n)}, \theta_{k,\ell}) + R(\boldsymbol{\Theta}) \qquad (1)$$

where $R(\boldsymbol{\Theta})$ is a regularization term (e.g., $\lambda||\boldsymbol{\Theta}||_2^2$ with $\lambda$ as a tuning parameter) (In Bayesian view, it is a prior on the parameters that could be informative or non-informative.) Let us use $J(\theta_{k,\ell}) = -\log P(y_{k,\ell}|x^{(n)}, \theta_{k,\ell})$ to indicate the cost of the data dependent term $\theta_{k,\ell}$. The exact form of $J(\theta_{k,\ell})$ would depend on the particular learning model being used over the labels $y$'s. For example, for logistic regression it would be $J(\theta_{k,\ell}) = -\log\left(\left(\frac{1}{1+e^{-\theta_{k,\ell}^T x^{(n)}}}\right)^{y_{k,\ell}} \left(1 - \frac{1}{1+e^{-\theta_{k,\ell}^T x^{(n)}}}\right)^{(1-y_{k,\ell})}\right)$.

Motivated by the earlier discussion, we want to model the interactions between the parameters of the different classification models, indexed by $\{k, \ell\}$ that we merge into one index $\{m\}$.

In this work, we represent these interactions as an undirected graph $\mathcal{G}$ where each node $m$ represents the parameters $\theta_m$. The edges $\mathcal{E}$ in the this graph would represent the interaction between two sets of parameters $\theta_i$ and $\theta_j$. These interactions are often sparse. We call this graph $\theta$-MRF. Eq. 1 can now be viewed as optimizing the energy function of the MRF *over the parameters*. I.e.,

$$\underset{\boldsymbol{\Theta}}{\text{minimize}} \sum_{m \in \mathcal{G}} J(\theta_m) + \sum_{i,j \in \mathcal{E}} R(\theta_i, \theta_j) \qquad (2)$$

where $J(\theta_m)$ is now the node potential, and the term $R(\theta_i, \theta_j)$ corresponds to the edge potentials. Note this idea of MRF is quite complementary of other modeling structures one may impose over $y$'s—which may itself be an MRF. This $\theta$-MRF is different from the label-based MRFs whose variables $y$'s are often in low-dimension. In our parameter-based MRF, each node constitutes high-dimensional variables $\theta_m$. One nice property of having an MRF over parameters is that there is no increase in complexity of the inference problem.

In previous work (also see Section 2), several priors have been used on the parameters. Such priors are often in the form of imposing a distribution with some other hyper parameters—this corresponds to a directed model on the $\boldsymbol{\Theta}$ and in some application scenarios they may not be able to express the desired conditional independence properties and therefore may be sub-optimal. Our $\theta$-MRF is

largely a non-informative prior, and also corresponds to some regularization methods. See Section 5 for experimental comparisons with different forms of priors. Having presented this general notion of $\theta$-MRF, we will now describe two types of interactions that it models well in the following.

**Spatial interactions.** Intuitively the parameters of the classifiers at neighboring spatial regions (for the same object category) should share their parameters. To model this type of interactions between parameters, we introduce edges on the $\theta$-MRF that connect the spatially neighboring nodes, as shown in Figure 1-left. Note that the spatial edges only couple the parameters of the same task together. This type of edge does not exist across tasks. We define the edge potential as follows.

$$R(\theta_i, \theta_j) = \begin{cases} \lambda_{\text{spt}} \|\theta_i - \theta_j\|_p & \text{if } \theta_i \text{ and } \theta_j \text{ are spatial neighbors for a task} \\ 0 & \text{otherwise} \end{cases}$$

where $\lambda_{\text{spt}}$ is a tuning factor for the spatial interactions. When $p \geq 1$, this potential has the nice property of being convex. Note that such a potential has been extensively used in an MRF over labels, e.g., [44]. Note that this potential does not make the original learning problem in Equation 1 any "harder." In fact, if the original objective $J(\theta)$ is convex, then the overall problem still remains convex. In this work, we consider $p = 1$ and $p = 2$.

In addition to connecting the parameters for neighboring locations, we also encourage the sharing between the elements of a parameter vector that correspond to spatially neighboring *inputs*. The intuition is described in the following example. Assume we have the presence of the object "road" at the different regions of an image as attributes. In order to learn a car detector with these attributes as inputs, we would like to give similar high-weights to the neighboring regions in the car detector output. We call this **source-based spatial grouping**, as compared to **target-based spatial grouping** that we described in the previous paragraph. We found that this also gives us a contextual map (i.e., parameters that map the feature/attributes in the neighboring regions) that is more spatially structured. This interaction happens within the same node in the graph, therefore it is equivalent to adding an extra term to the node potential on the $\theta$-MRF.

$$J_{\text{new}}(\theta_m) = J(\theta_m) + \lambda_{\text{src}} \sum_{t_1} \sum_{t_2 \in Nr(t_1)} \|\theta_m^{t_1} - \theta_m^{t_2}\|_p \qquad (3)$$

where $\theta_m^{t_1}$ and $\theta_m^{t_2}$ corresponds the weights given to the $t_1^{th}$ and the $t_2^{th}$ feature inputs. $t_2 \in Nr(t_1)$ means that the respective features are the same type of attributes form neighboring regions. Equation 3 can be reformed as $J_{\text{new}}(\theta_m) = J(\theta_m) + \lambda_{\text{src}} \|\mathcal{T}\theta_m\|_p$, where $\mathcal{T}$ indicates the linear transform matrix that computes the difference in the neighbors. $\lambda_{\text{src}}$ is a tuning factor for the source interactions.

**Semantic interactions.** We not only connect the parameters for spatial neighbors of the same task, but also consider the semantic neighbors across tasks. Motivated by the conditional independency in the object labels which suggests that given the important context the presence of an object is independent of other non-contextual objects, we can encode such properties in our $\theta$-MRF. For example, the road often appears below the car. Note that in our framework we have the road classifier and the car classifier take the same features as input, which are extracted from all regions of the images to capture long-range context. Since the high concurrence of these two objects, their corresponding detectors should be activated simultaneously. Therefore, the parameter for detecting "road" at a bottom region of the image, can partly share with the parameter for detecting "car" above the bottom region. Assume we already know the dependency between the objects, we introduce the semantic edge potential of the $\theta$-MRF, as shown in Figure 1-right.

$$R(\theta_i, \theta_j) = \begin{cases} \lambda_{\text{smn}} w_{ij} \|\theta_i - \theta_j\|_p & \text{if } \theta_i \text{ and } \theta_j \text{ are semantic neighbors} \\ 0 & \text{otherwise} \end{cases}$$

where $w_{ij}$ indicates the strength of the semantic dependency between these two parameters and $\lambda_{smn}$ is a tuning factor for the semantic interactions. In the following we discuss how to find the semantic connections and the weights $w$'s.

**Finding the semantic neighbors**. We first calculate the positive correlations between the tasks from the ground-truth training data. If two tasks are highly positively correlated, they are likely to share some of the parameters. In order to model how they share parameters, we model the relative spatial relationship between the positive outputs of the two tasks. For example, assume we have two highly co-occuring object categories, indexed by $k_1$ and $k_2$. From the training data, we learn the relative spatial distribution map of the presence of the $k_2^{th}$ object, given the $k_1^{th}$ object in the center. We then find out the top $M$ highest response regions on the map, each of which has a relative location $\Delta \ell$

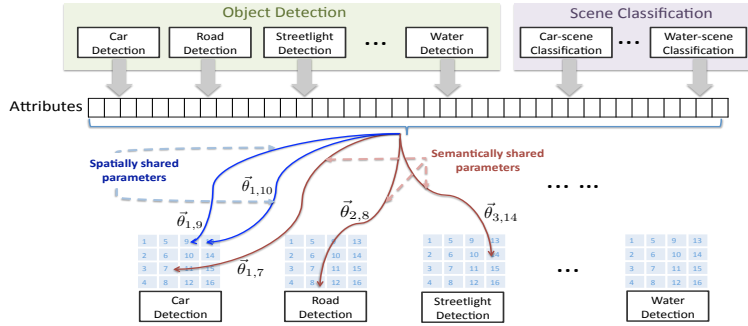

Figure 2: An instantiation of the proposed algorithm for the object recognition tasks on SUN09 dataset.

and co-occuring response $w$. Therefore, the parameters of the $k_2^{th}$ object that satisfy these relative locations, have semantic edges with $\theta_{k_1,l_1}$.

**Learning and Optimization.** $R(\mathbf{\Theta})$ couples the different independent parameters. Typically, the total number of parameters is quite large in an application (e.g., $47.6$ million in one of our applications, see Section 4). Running an optimization algorithm jointly on all the parameters would either not be feasible or have very slow convergence in practice. Since the parameters follow conditional independence assumptions and also follow a nice topological structure, we can optimize more connected subsets of the parameters separately, and then iterate. These separate sub-problems can also run in parallel. In our implementation, $R(\mathbf{\Theta})$'s and $J(\theta_m)$ are convex, and such a decomposed algorithm for optimizing the parameters is guaranteed to converge to the global optima [5].

## 4  Applications

We apply our $\theta$-MRF on two different settings: 1) object detection on the SUN09 dataset [7]; 2) multiple scene understanding tasks (scene categorization, geometric labeling, depth estimation), comparing to the cascaded classification models (CCM) [22, 31].

**Object Detection.** The task of object detection is to recognize and localize objects of interest in an image. We use the SUN 09 dataset introduced in [7], which has 4,367 training images and 4,317 test images. Choi et al. [7] use an additional set of 26,000 images to training baseline detectors [13], and select 107 object categories to evaluate their contextual model. We follow the *same settings* as [7], i.e., we use the same baseline object detector outputs as the attribute inputs for our algorithm, the same training/testing data, and the same evaluation metrics. For evaluation, a predicted bounding box is considered correct if it overlaps the ground-truth bounding box (in the intersection/union sense) by more than 50%. We compute the average precision (AP) of the precision-recall curve for each category, and compute the mean AP across categories as the overall performance.

We use each of the baseline object detectors to produce a $8 \times 8$ detection map, with each element indicating the confidence (between 0 and 1) of the object's presence at the respective region. We also define 107 scene categories, where the $i^{th}(i = 1, \ldots, 107)$ scene category indicates the type of scene containing the $i^{th}$ object category. We train a logistic regression classifier for each scene category. The 107 $8 \times 8$ object maps and the 107 scene classifier outputs together form a 6955-dimension feature vector, as the attribute inputs for our algorithm. The setup is shown in Figure 2.

We divide an image into $8 \times 8$ regions. Our algorithm learns a region-specific contextual model for each object category, resulting in a specific classifier of each region for each category. The $8 \times 8$ division is determined based on the criteria that more than 70% of the training data contain bounding boxes no smaller than a single grid. We use a linear model for each classifier. So we have $6955 * 8 * 8 * 107 = 47627840$ parameter dimensions in total. Our $\theta$-MRF captures the independencies between these parameters based on location and semantics. For the $l^{th}$ region, it is labeled as positive for the $k^{th}$ object category if it satisfies: $\text{overlap}(O_k, R_l)/\min\big(\text{area}(R_l), \text{area}(O_k)\big) > 0.3$, where $O_k$ means a bounding-box instantiation of the $k^{th}$ object category and $R_l$ means the $l^{th}$ grid cell. Negative examples are sampled from the false positives of the baseline detectors. We apply the trained classifiers to the test images, and gain the object detection maps. To create bounding-box based results, we use the candidate bounding boxes created by the baseline detectors, and average the scores gained from our algorithm within the bounding box as the confidence score for the candidate.

**Multiple Scene Understanding Tasks.** We consider the task of estimating different types of labels in a scene: scene categorization, geometry labeling, and depth estimation. We compose these three tasks in the feed-forward cascaded classification models (CCM) [22]. CCM creates repeated instantiations of each classifier on multiple layers of a cascade, where the latter-layer classifiers take the outputs of the previous-layer classifiers as input. The previous CCM algorithms [22, 31] consider sharing information across tasks, but do not consider the sharing between categories or between

Table 1: Performance of object recognition and detection on SUN09 dataset.

| Model | Object Recognition (% AP) | Object Detection (% AP) |
|---|---|---|
| Chance | 5.34 | N/A |
| Baseline (w/o context) | 17.9 | 7.06 |
| Single model per object | 22.3 | 8.02 |
| Independent model | 22.9 | 8.18 |
| State-of-the-art [7] | 25.2 | 8.33 |
| $\theta$-**MRF** ($l_2$-**regularized**) | **26.4** | **8.76** |
| $\theta$-**MRF** ($l_1$-**regularized**) | **27.0** | **8.93** |

Table 2: Performance of scene categorization, geometric labeling, and depth estimation in CCM.

| Model | Scene Categorization (% AP) | Geometric Labeling (% AP) | Depth Estimation (RMSE in m) |
|---|---|---|---|
| Chance | 22.5 | 33.3 | 24.6 |
| Baseline(w/o context) | 83.8 | 86.2 | 16.7 |
| State-of-the-art [31] | 86.1 | 88.9 | 15.2 |
| CCM [22] (our implementation) | 83.8 | 87.0 | 16.5 |
| $\theta$-**MRF** ($l_2$-**regularized**) | **85.7** | **88.6** | **15.3** |
| $\theta$-**MRF** ($l_1$-**regularized**) | **86.3** | **89.2** | **15.2** |

different spatial regions within a task. Here we introduce the semantically-grouped regularization to scene categorization, and the spatially-grouped regularization to depth and geometry estimation.

For the three tasks we consider, we use the same datasets and 2-layer settings as [31]. For scene categorization, we classify 8 different categories on the MIT outdoor scene dataset [39]. We consider two semantic groups: man-made (tall building, inside city, street, highway) and natural (coast, open-country, mountain and forest). Semantic edges are introduced between the parameters within each group. We train a logistic classifier for each scene category. This gives us a total of 8 parameter vectors for scene categorization task. We evaluate the performance by measuring the accuracy of assigning the correct scene label to an image.

For depth estimation, we train a specific linear regression model for every region of the image (with uniformly divided $11 \times 10$ regions), and incorporate the spatial grouping on both the second-layer inputs and outputs. This gives us a total of 110 parameter vectors for the depth estimation task. We evaluate the performance by computing the root mean square error of the estimated depth with respect to ground truth laser scan depth using the Make3D Range Image dataset [44].

For geometry labeling, We use the dataset and the algorithm by [24] as the first-layer geometric labeling module, and use a single segmentation with about 100 segments/image. On the second-layer, we train a logistic regression classifier for every region of the image (with uniformly divided $16 \times 16$ regions), and incorporate the spatial grouping on both the second-layer inputs and outputs. This gives us a total of 768 parameter vectors. We then assign the geometric label to each segment based on the average confidence scores within the segment. We evaluate the performance by computing the accuracy of assigning the correct geometric label to a pixel.

## 5 Experiments

We evaluate the proposed algorithm on two applications: (1) object recognition and detection on SUN09 dataset with 107 object categories; (2) the multi-task cascaded structure that composes scene categorization, depth estimation and geometric labeling on multiple datasets as described in Section 4. The training of our algorithm takes 6-7 hours for object detection/recognition and 3-4 hours for multi-task cascade. The attribute models in (1) and the first-layer base classifiers in (2) are pre-trained. The complexity of our inference is no more than constant times of the complexity of inference of an individual classifier. Furthermore, the inference for different classifiers can be easily parallelized. For example, a base object detector [13] takes about 1.5 second to output results for an image. Our algorithm, taking the outputs of the base detectors as input, only requires an overhead of less than 0.2 second.

### 5.1 Overall performance on multiple tasks in CCM strcuture.

Table 2 shows the performance of different methods on the three tasks composed into the cascaded classification model (CCM) [22]. "Baseline" means the individual classifier for each task on the first layer, "State-of-the-art" corresponds to the state-of-the-art algorithm for each sub-task respectively for that specic dataset, and "CCM" corresponds to the second-layer output for each sub-task in the CCM structure. The results are computed as the average performance over 6-fold cross validation. With the semantic and spatial regularization, our proposed $\theta$-MRF algorithm improves significantly over the CCM algorithm that also uses the same set of tasks for prediction. Finally, we perform better than the state-of-the-art algorithms on two tasks and comparably for the third.

**Is $\theta$-MRF "complementary" to label-MRF?** In this experiment, we also consider the MRF over labels [44] together with our $\theta$-MRF for depth estimation. The combination results in a lower root-mean-square-error (RMSE) of 15.0m as compared to 15.2m for $\theta$-MRF alone and 16.0m for label-MRF alone. This indicates that our method is complementary to the traditional MRF over labels.

### 5.2 Overall performance on SUN09 object detection.

Table 1 gives the performance of different methods on SUN09 dataset, for both object recognition (predicting the object presence) and object detection (predicting the object location).

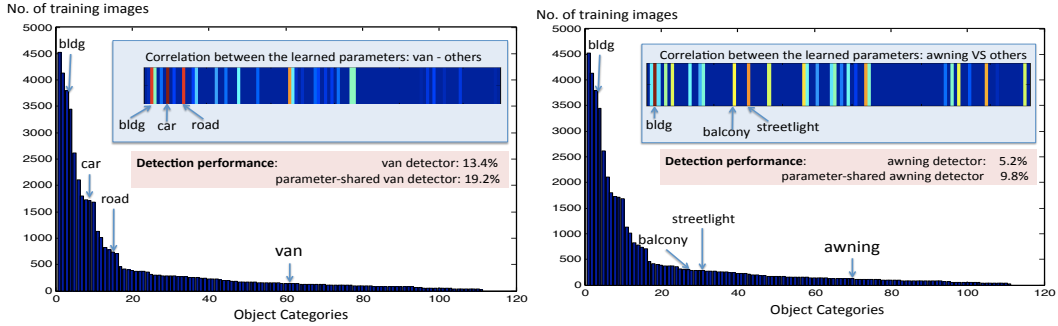

Figure 3: Examples showing that infrequent object categories share parameters with frequent object categories.

- Baseline (w/o context): the baseline object detectors trained by [13], which are also used to generate the initial detection results used as inputs for our algorithm and the state-of-the-art algorithm.

- Single model: a single classifier is trained for each object category, not varying across different locations. In the following, if not specified, we use a $l_1$-regularized linear regression as the classifier.

- Independent model: this means an independent classifier is trained for the presence of an object for each region. There is no information sharing between the models belonging to different locations of the same category, or different categories.

- State-of-the-art: This is the tree-based graphical model proposed in [7], which explicitly models the object dependencies based on labels and detector outputs.[2]

- The proposed $\theta$-MRF algorithm, which shares the models spatially within an object category and semantically across various objects. We evaluate both the $l_1$ and $l_2$ regularization on the potentials.

Table 1 shows the location-specific model (Independent) is better than the general model (Single model), which confirms our intuition that the contextual model is location-specific. Furthermore, our approach that shares parameters spatially and semantically significantly outperforms the independent model without these regularizations. We also note that our algorithm can achieve comparable performance to the state-of-the-art algorithm, without explicitly modeling the probabilistic dependency between the objects labels.

We study the relative improvement of the proposed parameter sharing algorithm over the non-parameter-sharing algorithm (Independent model in Table 1) on object categories with different number of training samples in the SUN09 object recognition task. The relative improvement on object categories with less than 200 training samples is 34.2%, while the improvement on objects with more than 200 training samples is 11.5%. Our parameter sharing algorithm helps the infrequent objects implicitly make use of the data of frequent objects to learn better models.

We give two examples in Fig. 3, focusing on two infrequent object categories: van and awning, respectively. The histogram in the figures shows the number of training instances for each object category. The color bar shows the correlation between the learned parameter of the object with the parameters for other objects. The redder indicates the higher correlation between the parameters of the respective categories. Figure 3-left shows that the van category has few training instances, turn out to share the parameters strongly with the categories of car, building and road. Similarly, Figure 3-right shows how the learned awning parameters with other categories. We note that in the dataset, awning and streetlight are not highly co-occuring, thus initially when we create the semantic groups, these two objects *do not* appear simultaneously in any group. However, the semantic groups containing streetlight and the semantic groups containing awning both contain objects like road, building, and car. Through our $\theta$-MRF algorithm, the sharing information can be transferred.

**Effect of different priors.** We compare our spatially-grouped and semantically-grouped regularization with other parameter sharing algorithms such as the prior-based algorithms in Figure 4.

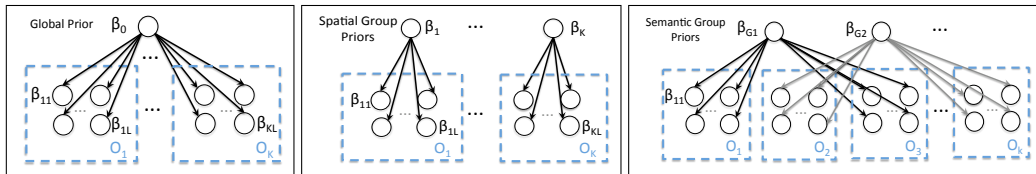

Figure 4: Some baseline prior-based algorithms we compare the propose algorithm with. From left to right: these models use global prior, spatial-based prior, and semantic-based prior.

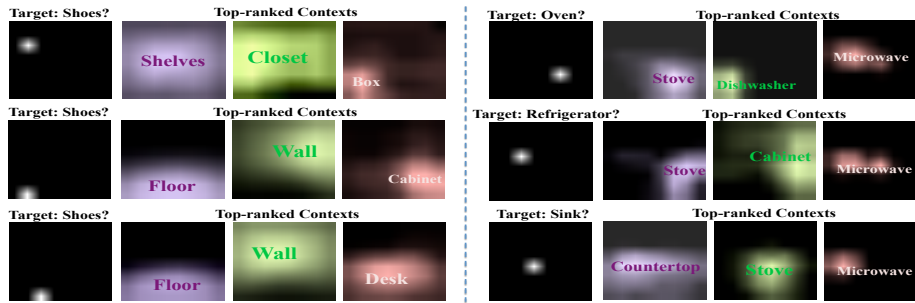

Figure 5: Examples of the visual context learned from the proposed algorithm. Six examples are given. In each example, the left figure illustrates the task: whether the white region belongs to the target category. The following three figures shows the contextual inputs (showing the spatial map) which have the top ranked weights (highest positive elements of the parameters) .

- Global prior (Fig. 4-left): all the classifiers share the prior $\beta_0$, and the parameter for a classifier is defined as: $\theta_{k,l} = \beta_0 + \beta_{k,l}$. Assuming zero-mean gaussian distribution for each $\beta$, we have the regularization term in Equation 1: $R(\Theta) = \sum_{k,l} R(\theta_{k,l}) = \sum_{k,l} \left( \lambda_0 \|\beta_0\|_2 + \lambda_{k,l} \|\beta_{k,l}\|_2 \right)$.
- Spatial prior (Fig. 4-middle): the classifiers for the same object $O_k$ at different locations share a prior $\theta_k$,i.e.,$\theta_{k,l} = \beta_k + \beta_{k,l}$. Thus we have $R(\Theta) = \sum_{k,l} R(\theta_{k,l}) = \sum_{k,l} \left( \lambda_k \|\beta_k\|_2 + \lambda_{k,l} \|\beta_{k,l}\|_2 \right)$.
- Semantic prior (Fig. 4-right): the classifiers from the same semantic group share a prior $\beta_{G_i}$, and the parameter for a classifier is defined as: $\theta_{k,l} = \beta_{G_i} + \beta_{k,l}$, where $\theta_{k,l} \in G_i$. Thus we have $R(\Theta) = \sum_{k,l} R(\theta_{k,l}) = \sum_{k,l} \left( \lambda_{G_i} \|\beta_{G_i}\|_2 + \lambda_{k,l} \|\beta_{k,l}\|_2 \right)$. The semantic groups are generated by an agglomerative clustering based on the co-occurence of objects.
- Spatial $\theta$-MRF and Semantic $\theta$-MRF (Fig. 1): the proposed regularizations in Section 3 and $l-2$ norm is used as the regularization form.

Table 3 shows that the proposed $\theta$-MRF algorithms outperform the prior-based algorithms in both the spatial-grouping and semantic-grouping settings. Sharing the only global prior across all tasks performs slightly better than the independent $l_2$ regularization based classifier. Modeling the spatial and semantic interactions by both methods (adding priors or adding edges) improve the performance, while the $\theta$-MRF based approach is more effective, especially with $l_1$ norm.

**Visual grouping.** Figure 5 illustrates the effect of our proposed parameter sharing. For each object in a location, we show the top three

Table 3: Results for different parameter sharing methods on object recognition task.

| Models | Object Recog. (% AP) |
|---|---|
| No prior, $l_2$-sparsity | 22.5 |
| Global prior | 22.8 |
| Spatial priors | 23.6 |
| **Spatial $\theta$-MRF** | **24.6** |
| Semantic priors | 24.0 |
| **Semantic $\theta$-MRF** | **25.2** |
| **Full $\theta$-MRF** | **26.4** |
| **Full $\theta$-MRF, $l_1$** | **27.0** |

contextual inputs learned by our approach. In Figure 5-left, we show where the highest positive weights locate in order to detect shoes at different regions. We note that to detect shoes in topper part of the image, shelves, closet and box are the most important contextual inputs; while floor and wall play a more important role in detecting shoes at the bottom of the image. The results also show that the two neighboring shoe regions (row 2 and row 3) share similar context, while far-away ones (row 1) do not. This reflects our target-based spatial interaction within the parameter-MRF.

In Figure 5-right, we show a group of parameters (corresponding to different regions for oven, refrigerator, and sink) that share semantic edges with each other on the parameter-MRF. We note that they share the high weights given to stove at the bottom-right and microwave at the middle-left. All these objects have very few training examples. With the proposed semantic constraint, they can implicitly leverage information from each other in the training stage. Besides we also note the spatially smooth effect on the figures, which is resulted from our source-based spatial interactions.

**Conclusion.** We propose a method to capture structure in the parameters by designing an *MRF over parameters*. Our evaluations show that our method performs better than the current state-of-the-art algorithms on four different tasks (that were specifically designed for the respective tasks). Note that our method is complementary to the techniques state-of-the-art methods use for the respective tasks (e.g., MRF on the labels for depth estimation), and we believe that one can get even higher performance by combining our $\theta$-MRF technique with the respective state-of-the-art techniques.

## Footnotes

[1]As an example, consider the problem of object detection with many categories: we have 107 object cate-

[2]We evaluate the contextual model in [7] using the software published by the authors: `http://web.mit.edu/~myungjin/www/HContext.html` and report the average performance on multiple runs.

# References

[1] S. Bengio, F. Pereira, and Y. Singer. Group sparse coding. In *NIPS*, 2009.

[2] M. Blaschko and C. Lampert. Object localization with global and local context kernels. In *BMVC*, 2009.

[3] D. M. Blei, A. Y. Ng, and M. I. Jordan. Latent dirichlet allocation. *JMLR*, 3:993–1022, 2003.

[4] G. E. Box and G. C. Tiao. *Bayesian Inference in Statistical Analysis*. John Wiley & Sons, 1992.

[5] S. Boyd and L. Vandenberghe. *Convex Optimization*. Cambridge [u.a.]: Univ. Press, 2004.

[6] C. Brezinski, M. Redivo-Zaglia, G. Rodriguez, and S. Seatzu. Multi-parameter regularization techniques for ill-conditioned linear systems. *Mathematics and Statistics*, 94(2):203–228, 2003.

[7] M. J. Choi, J. J. Lim, A. Torralba, and A. S. Willsky. Exploiting hierarchical context on a large database of object categories. In *CVPR*, 2010.

[8] A. Dawid and S. Lauritzen. Hyper markov laws in the statistical analysis of decomposable graphical models. *The Annals of Statistics*, 1993.

[9] C. Desai, D. Ramanan, and C. Fowlkes. Discriminative models for multi-class object layout. In *ICCV'09*.

[10] S. Divvala, D. Hoiem, J. Hays, A. Efros, and M. Hebert. An empirical study of context in object detection. In *CVPR*, 2009.

[11] C. B. Do, C.-S. Foo, and A. Y. Ng. Efficient multiple hyperparameter learning for log-linear models. In *NIPS*, 2007.

[12] L. Fei-Fei, R. Fergus, and P. Perona. A bayesian approach to unsupervised one-shot learning of object categories. In *CVPR*, 2003.

[13] P. F. Felzenszwalb, R. B. Girshick, D. McAllester, and D. Ramanan. Object detection with discriminatively trained part based models. *PAMI*, 2009.

[14] R. Fergus, H. Bernal, Y. Weiss, and A. Torralba. Semantic label sharing for learning with many categories. In *ECCV*, 2010.

[15] R. Fergus, Y. Weiss, and A. Torralba. Semi-supervised learning in gigantic image collections. In *NIPS'09*.

[16] C. Galleguillos, B. McFee, S. Belongie, and G. Lanckriet. Multi-class object localization by combining local contextual interactions. In *CVPR*, 2010.

[17] C. Galleguillos, A. Rabinovich, and S. Belongie. Object categorization using co-occurrence, location and appearance. In *CVPR*, 2008.

[18] P. Garrigues and B. Olshausen. Group sparse coding with a laplacian scale mixture prior. In *NIPS*, 2010.

[19] S. Gould, J. Rodgers, D. Cohen, G. Elidan, and D. Koller. Multi-class segmentation with relative location prior. *IJCV*, 80(3), 2008.

[20] J. V. Graa, K. Ganchev, B. Taskar, and F. Pereira. Posterior vs. parameter sparsity in latent variable models. In *NIPS*, 2009.

[21] D. Heinz. Hyper markov non-parametric processes for mixture modeling and model selection. In *CMU*.

[22] G. Heitz, S. Gould, A. Saxena, and D. Koller. Cascaded classification models: Combining models for holistic scene understanding. In *NIPS*, 2008.

[23] G. Heitz and D. Koller. Learning spatial context: Using stuff to find things. In *ECCV*, 2008.

[24] D. Hoiem, A. A. Efros, and M. Hebert. Putting objects in perspective. *IJCV*, 2008.

[25] J. Huang, T. Zhang, and D. Metaxas. Learning with structured sparsity. In *ICML*, 2009.

[26] A. Jalali, P. Ravikumar, S. Sanghavi, and C. Ruan. A dirty model for multi-task learning. In *NIPS*, 2010.

[27] S. Kim and E. P. Xing. Tree-guided group lasso for multi-task regression with structured sparsity. In *ICML*, 2010.

[28] S. Kumar and M. Hebert. A hierarchical field framework for unified context-based classification. In *ICCV*, 2005.

[29] S. Kumar and Singh. Discriminative fields for modeling spatial dependencies in natural images. In *NIPS*, 2004.

[30] C. H. Lampert, H. Nickisch, and S. Harmeling. Learning to detect unseen object classes by between-class attribute transfer. In *CVPR*, 2009.

[31] C. Li, A. Kowdle, A. Saxena, and T. Chen. Feedback enabled cascaded classication models for scene understanding. In *NIPS*, 2010.

[32] L.-J. Li, R. Socher, and L. Fei-Fei. Towards total scene understanding: Classification, annotation and segmentation in an automatic framework. In *CVPR*, 2009.

[33] L.-J. Li, H. Su, E. P. Xing, and L. Fei-Fei. Object bank: A high-level image representation for scene classification and semantic feature sparsification. In *NIPS*, 2010.

[34] P. Liang, F. Bach, G. Bouchard, and M. I. Jordan. Asymptotically optimal regularization in smooth parametric models. In *NIPS*, 2010.

[35] J. Lim, P. Arbel anez, C. Gu, and J. Malik. Context by region ancestry. In *ICCV*, 2009.

[36] M. Marszalek and C. Schmid. Semantic hierarchies for visual object recognition. In *CVPR*, 2007.

[37] D. Munoz, J. Bagnell, and M. Hebert. Stacked hierarchical labeling. In *ECCV*, 2010.

[38] S. Negahban and M. J. Wainwright. Joint support recovery under high-dimensional scaling: Benefits and perils of l1,-regularization. In *NIPS*, 2008.

[39] A. Oliva and A. Torralba. Modeling the shape of the scene: A holistic representation of the spatial envelope. *IJCV*, 42:145–175, 2001.

[40] A. Rabinovich and et al. Objects in context. In *ICCV*, 2007.

[41] A. Roverato. Hyper inverse wishart distribution for non-decomposable graphs and its application to bayesian inference for gaussian graphical models. *Scandinavian Journal of Statistics*, 2002.

[42] R.Tibshirani. Regression shrinkage and selection via the lasso. *J Royal Stat. Soc. B*, 58(1):267–288, 1996.

[43] R. Salakhutdinov, A. Torralba, and J. Tenenbaum. Learning to share visual appearance for multiclass object detection. In *CVPR*, 2011.

[44] A. Saxena, S. H. Chung, and A. Y. Ng. 3-d depth reconstruction from a single still image. *IJCV*, 76, 2007.

[45] A. Saxena, M. Sun, and A. Y. Ng. Make3d: Learning 3d scene structure from a single still image. *IEEE PAMI*, 30(5), 2009.

[46] E. Sudderth, A. Torralba, W. Freeman, and A. Willsky. Learning hierarchical models of scenes, objects, and parts. In *ICCV*, 2005.

[47] A. Torralba. Contextual priming for object detection. *Int. J. Comput. Vision*, 53(2):169–191, 2003.

[48] B. Yao and L. Fei-Fei. Modeling mutual context of object and human pose in human-object interaction activities. In *CVPR*, 2010.

